# The Power of Amnesia

**Dana Ron    Yoram Singer    Naftali Tishby**
Institute of Computer Science and
Center for Neural Computation
Hebrew University, Jerusalem 91904, Israel

## Abstract

We propose a learning algorithm for a variable memory length Markov process. Human communication, whether given as text, handwriting, or speech, has multi characteristic time scales. On short scales it is characterized mostly by the dynamics that generate the process, whereas on large scales, more syntactic and semantic information is carried. For that reason the conventionally used fixed memory Markov models cannot capture effectively the complexity of such structures. On the other hand using long memory models uniformly is not practical even for as short memory as four. The algorithm we propose is based on minimizing the statistical prediction error by extending the memory, or state length, adaptively, until the total prediction error is sufficiently small. We demonstrate the algorithm by learning the structure of natural English text and applying the learned model to the correction of corrupted text. Using less than 3000 states the model's performance is far superior to that of fixed memory models with similar number of states. We also show how the algorithm can be applied to intergenic *E. coli* DNA base prediction with results comparable to HMM based methods.

## 1    Introduction

Methods for automatically acquiring the structure of the human language are attracting increasing attention. One of the main difficulties in modeling the natural language is its multiple temporal scales. As has been known for many years the language is far more complex than any finite memory Markov source. Yet Markov

models are powerful tools that capture the short scale statistical behavior of language, whereas long memory models are generally impossible to estimate. The obvious desired solution is a Markov source with a 'deep' memory just where it is really needed. Variable memory length Markov models have been in use for language modeling in speech recognition for some time [3, 4], yet no systematic derivation, nor rigorous analysis of such learning mechanism has been proposed.

Markov models are a natural candidate for language modeling and temporal pattern recognition, mostly due to their mathematical simplicity. It is nevertheless obvious that finite memory Markov models can not in any way capture the recursive nature of the language, nor can they be trained effectively with long enough memory. The notion of a variable length memory seems to appear naturally also in the context of universal coding [6]. This information theoretic notion is now known to be closely related to efficient modeling [7]. The natural measure that appears in information theory is the description length, as measured by the statistical predictability via the Kullback- Liebler (KL) divergence.

The algorithm we propose here is based on optimizing the statistical prediction of a Markov model, measured by the instantaneous KL divergence of the following symbols, or by the current *statistical surprise* of the model. The memory is extended precisely when such a surprise is significant, until the overall statistical prediction of the stochastic model is sufficiently good. We apply this algorithm successfully for statistical language modeling. Here we demonstrate its ability for spelling correction of corrupted English text. We also show how the algorithm can be applied to intergenic *E. coli* DNA base prediction with results comparable to HMM based methods.

## 2   Prediction Suffix Trees and Finite State Automata

### Definitions and Notations

Let $\Sigma$ be a finite alphabet. Denote by $\Sigma^\star$ the set of all strings over $\Sigma$. A string $s$, over $\Sigma^\star$ of length $n$, is denoted by $s = s_1 s_2 \ldots s_n$. We denote by $\mathbf{e}$ the empty string. The length of a string $s$ is denoted by $|s|$ and the size of an alphabet $\Sigma$ is denoted by $|\Sigma|$. Let, $Prefix(s) = s_1 s_2 \ldots s_{n-1}$, denote the longest prefix of a string $s$, and let $Prefix^\star(s)$ denote the set of all prefixes of $s$, including the empty string. Similarly, $Suffix(s) = s_2 s_3 \ldots s_n$ and $Suffix^\star(s)$ is the set of all suffixes of $s$. A set of strings is called a prefix free set if, $\forall s^1, s^2 \in S : \{s^1\} \bigcap Prefix^\star(s^2) = \emptyset$. We call a probability measure $P$, over the strings in $\Sigma^\star$ *proper* if $P(\mathbf{e}) = 1$, and for every string $s$, $\sum_{\sigma \in \Sigma} P(s\sigma) = P(s)$. Hence, for every prefix free set $S$, $\sum_{s \in S} P(s) \leq 1$, and specifically for every integer $n \geq 0$, $\sum_{s \in \Sigma^n} P(s) = 1$.

### Prediction Suffix Trees

A prediction suffix tree $T$ over $\Sigma$, is a tree of degree $|\Sigma|$. The edges of the tree are labeled by symbols from $\Sigma$, such that from every internal node there is at most one outgoing edge labeled by each symbol. The nodes of the tree are labeled by pairs $(s, \gamma_s)$ where $s$ is the string associated with the walk starting from that node and ending in the root of the tree, and $\gamma_s : \Sigma \longrightarrow [0, 1]$ is the *output probability function* related with $s$ satisfying $\sum_{\sigma \in \Sigma} \gamma_s(\sigma) = 1$. A prediction suffix tree induces

probabilities on arbitrary long strings in the following manner. The probability that $T$ generates a string $w = w_1 w_2 \ldots w_n$ in $\Sigma^n$, denoted by $P_T(w)$, is $\Pi_{i=1}^n \gamma_{s^{i-1}}(w_i)$, where $s^0 = \mathbf{e}$, and for $1 \le i \le n-1$, $s^j$ is the string labeling the *deepest* node reached by taking the walk corresponding to $w_1 \ldots w_i$ starting at the root of $T$. By definition, a prediction suffix tree induces a proper measure over $\Sigma^\star$, and hence for every prefix free set of strings $\{w^1, \ldots, w^m\}$, $\sum_{i=1}^m P_T(w^i) \le 1$, and specifically for $n \ge 1$, then $\sum_{s \in \Sigma^n} P_T(s) = 1$. An example of a prediction suffix tree is depicted in Fig. 1 on the left, where the nodes of the tree are labeled by the corresponding suffix they present.

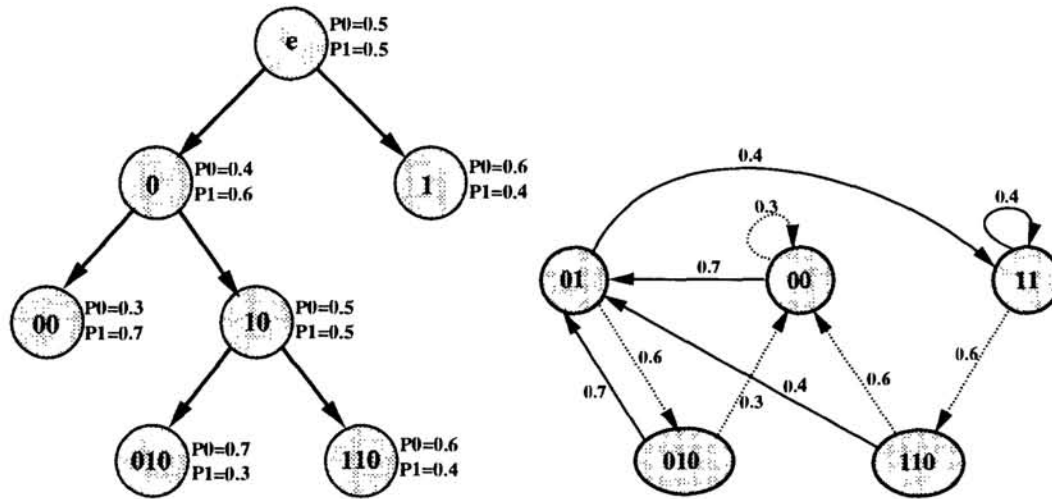

Figure 1: Right: A prediction suffix tree over $\Sigma = \{0, 1\}$. The strings written in the nodes are the suffixes the nodes present. For each node there is a probability vector over the next possible symbols. For example, the probability of observing a '1' after observing the string '010' is 0.3. Left: The equivalent probabilistic finite automaton. Bold edges denote transitions with the symbol '1' and dashed edges denote transitions with '0'. The states of the automaton are the leaves of the tree except for the leaf denoted by the string **1**, which was replaced by the prefixes of the strings **010** and **110**: **01** and **11**.

**Finite State Automata and Markov Processes**

A *Probabilistic Finite Automaton (PFA)* $A$ is a 5-tuple $(Q, \Sigma, \tau, \gamma, \pi)$, where $Q$ is a finite set of $n$ *states*, $\Sigma$ is an *alphabet* of size $k$, $\tau : Q \times \Sigma \rightarrow Q$ is the *transition function*, $\gamma : Q \times \Sigma \rightarrow [0, 1]$ is the *output probability function*, and $\pi : Q \rightarrow [0, 1]$ is the probability distribution over the *starting states*. The functions $\gamma$ and $\pi$ must satisfy the following requirements: for every $q \in Q$, $\sum_{\sigma \in \Sigma} \gamma(q, \sigma) = 1$, and $\sum_{q \in Q} \pi(q) = 1$. The probability that $A$ generates a string $s = s_1 s_2 \ldots s_n \in \Sigma^n$ is $P_A(s) = \sum_{q^0 \in Q} \pi(q^0) \prod_{i=1}^n \gamma(q^{i-1}, s_i)$, where $q^{i+1} = \tau(q^i, s_i)$.

We are interested in learning a sub-class of finite state machines which have the following property. Each state in a machine $M$ belonging to this sub-class is labeled by a string of length at most $L$ over $\Sigma$, for some $L \ge 0$. The set of strings labeling the states is suffix free. We require that for every two states $q^1, q^2 \in Q$ and for every symbol $\sigma \in \Sigma$, if $\tau(q^1, \sigma) = q^2$ and $q^1$ is labeled by a string $s^1$, then $q^2$ is labeled

by a string $s^2$ which is a suffix of $s^1 \cdot \sigma$. Since the set of strings labeling the states is suffix free, if there exists a string having this property then it is unique. Thus, in order that $\tau$ be well defined on a given set of string $S$, not only must the set be suffix free, but it must also have the property, that for every string $s$ in the set and every symbol $\sigma$, there exists a string which is a suffix of $s\sigma$. For our convenience, from this point on, if $q$ is a state in $Q$ then $q$ will also denote the string labeling that state.

A special case of these automata is the case in which $Q$ includes *all* $2^L$ strings of length $L$. These automata are known as *Markov processes of order $L$*. We are interested in learning automata for which the number of states, $n$, is actually much smaller than $2^L$, which means that few states have "long memory" and most states have a short one. We refer to these automata as Markov processes *with bounded memory $L$*. In the case of Markov processes of order $L$, the "identity" of the states (i.e. the strings labeling the states) is known and learning such a process reduces to approximating the output probability function. When learning Markov processes with bounded memory, the task of a learning algorithm is much more involved since it must reveal the identity of the states as well.

It can be shown that under a slightly more complicated definition of prediction suffix trees, and assuming that the initial distribution on the states is the stationary distribution, these two models are equivalent up to a grow up in size which is at most linear in $L$. The proof of this equivalence is beyond the scope of this paper, yet the transformation from a prediction suffix tree to a finite state automaton is rather simple. Roughly speaking, in order to implement a prediction suffix tree by a finite state automaton we define the leaves of the tree to be the states of the automaton. If the transition function of the automaton, $\tau(\cdot, \cdot)$, can not be well defined on this set of strings, we might need to slightly expand the tree and use the leaves of the expanded tree. The output probability function of the automaton, $\gamma(\cdot, \cdot)$, is defined based on the prediction values of the leaves of the tree. i.e., for every state (leaf) $s$, and every symbol $\sigma$, $\gamma(s, \sigma) = \gamma_s(\sigma)$. The outgoing edges from the states are defined as follows: $\tau(q^1, \sigma) = q^2$ where $q^2 \in Suffix^\star(q^1\sigma)$. An example of a finite state automaton which corresponds to the prediction tree depicted in Fig. 1 on the left, is depicted on the right part of the figure.

## 3    Learning Prediction Suffix Trees

Given a sample consisting of one sequence of length $l$ or $m$ sequences of lengths $l_1, l_2, \ldots, l_m$ we would like to find a prediction suffix tree that will have the same statistical properties of the sample and thus can be used to predict the next outcome for sequences generated by the same source. At each stage we can transform the tree into a Markov process with bounded memory. Hence, if the sequence was created by a Markov process, the algorithm will find the structure and estimate the probabilities of the process. The key idea is to iteratively build a prediction tree whose probability measure equals the empirical probability measure calculated from the sample.

We start with a tree consisting of a single node (labeled by the empty string **e**) and add nodes which we have reason to believe should be in the tree. A node $\sigma s$, must be added to the tree if it statistically differs from its parent node $s$. A natural measure

to check the statistical difference is the *relative entropy* (also known as the Kullback-Liebler (KL) divergence) [5], between the conditional probabilities $P(\cdot|s)$ and $P(\cdot|\sigma s)$. Let $X$ be an observation space and $P_1, P_2$ be probability measures over $X$ then the KL divergence between $P_1$ and $P_2$ is, $D_{KL}(P_1||P_2) = \sum_{x \in X} P_1(x) \log \frac{P_1(x)}{P_2(x)}$. Note that this distance is not symmetric and $P_1$ should be absolutely continuous with respect to $P_2$. In our problem, the KL divergence measures how much additional information is gained by using the suffix $\sigma s$ for prediction instead of predicting using the shorter suffix $s$. There are cases where the statistical difference is large yet the probability of observing the suffix $\sigma s$ itself is so small that we can neglect those cases. Hence we weigh the the statistical error by the prior probability of observing $\sigma s$. The statistical error measure in our case is,

$$
\begin{aligned}
Err(\sigma s, s) &= P(\sigma s)\, D_{KL}\left(P(\cdot|\sigma s)||P(\cdot|s)\right) \\
&= P(\sigma s) \sum_{\sigma' \in \Sigma} P(\sigma'|\sigma s) \log \frac{P(\sigma'|\sigma s)}{P(\sigma'|s)} \\
&= \sum_{\sigma' \in \Sigma} P(\sigma s \sigma') \log \frac{P(\sigma s \sigma')}{P(\sigma'|s)P(\sigma s)} \quad .
\end{aligned}
$$

Therefore, a node $\sigma s$ is added to the tree if the statistical difference (defined by $Err(\sigma s, s)$) between the node and its parrent $s$ is larger than a predetermined accuracy $\epsilon$. The tree is grown level by level, adding a son of a given leaf in the tree whenever the statistical surprise is large. The problem is that the requirement that a node statistically differs from it's parent node is a necessary condition for belonging to the tree, but is not sufficient. The leaves of a prediction suffix tree must differ from their parents (or they are redundant) but internal nodes might not have this property. Therefore, we must continue testing further potential descendants of the leaves in the tree up to depth $L$. In order to avoid exponential grow in the number of strings tested, we do not test strings which belong to branches which are reached with small probability. The set of strings, tested at each step, is denoted by $S$, and can be viewed as a kind of potential 'frontier' of the growing tree $T$. At each stage or when the construction is completed we can produce the equivalent Markov process with bounded memory. The learning algorithm of the prediction suffix tree is depicted in Fig. 2. The algorithm gets two parameters: an accuracy parameter $\epsilon$ and the maximal order of the process (which is also the maximal depth of the tree) $L$.

The true source probabilities are not known, hence they should be estimated from the empirical counts of their appearances in the observation sequences. Denote by $\#s$ the number of time the string $s$ appeared in the observation sequences and by $\#\sigma|s$ the number of time the symbol $\sigma$ appeared after the string $s$. Then, using Laplace's rule of succession, the empirical estimation of the probabilities is,

$$
P(s) \approx \tilde{P}(s) \triangleq \frac{\#s + 1}{\sum_{s' \in \Sigma^{|s|}} \#s' + |\Sigma|} \qquad P(\sigma|s) \approx \tilde{P}(\sigma|s) \triangleq \frac{\#\sigma|s + 1}{\sum_{\sigma' \in \Sigma} \#\sigma'|s + |\Sigma|}
$$

## 4  A Toy Learning Example

The algorithm was applied to a 1000 symbols long sequence produced by the automaton depicted top left in Fig. 3. The alphabet was binary. Bold lines in the figure represent transition with the symbol '0' and dashed lines represent the symbol '1'. The prediction suffix tree is plotted at each stage of the algorithm. At the

- Initialize the tree $T$ and the candidate strings $S$:
  $T$ consists of a single root node, and $S \leftarrow \{\sigma \mid \sigma \in \Sigma \land \tilde{P}(\sigma) \geq \epsilon\}$.
- While $S \neq \emptyset$, do the following:
  1. Pick any $s \in S$ and remove it from $S$.
  2. If $Err(s, Suffix(s)) \geq \epsilon$ then add to $T$ the node corresponding to $s$ and all the nodes on the path from the deepest node in $T$ (the deepest ancestor of $s$) until $s$.
  3. If $|s| < L$ then for every $\sigma \in \Sigma$ if $\tilde{P}(\sigma s) \geq \epsilon$ add $\sigma s$ to $S$.

Figure 2: The algorithm for learning a prediction suffix tree.

end of the run the correponding automaton is plotted as well (bottom right). Note that the original automaton and the learned automaton are the same except for small diffrences in the transition probabilities.

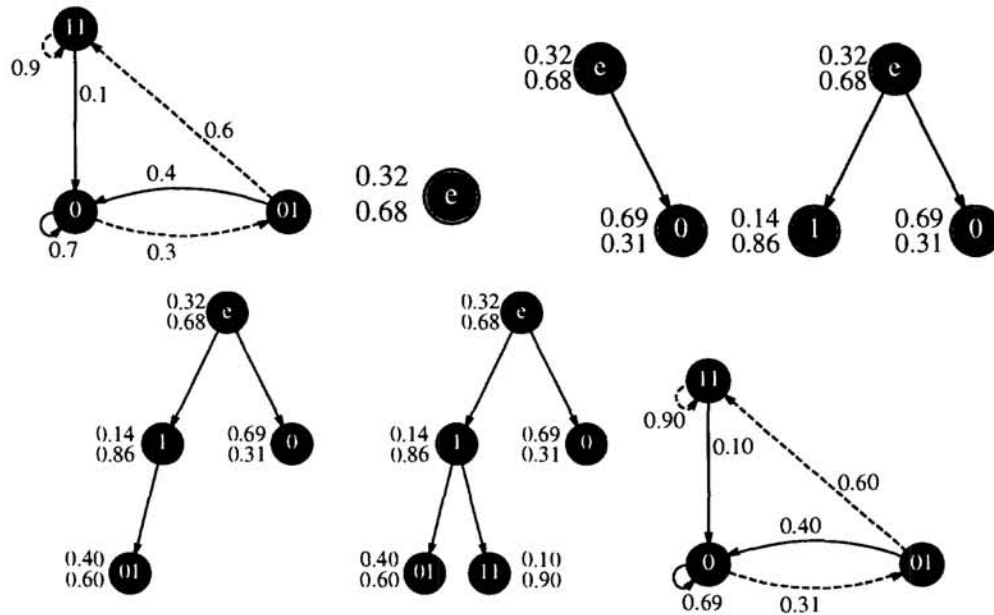

Figure 3: The original automaton (top left), the instantaneous automata built along the run of the algorithm (left to right and top to bottom), and the final automaton (bottom left).

## 5    Applications

We applied the algorithm to the Bible with $L = 30$ and $\epsilon = 0.001$ which resulted in an automaton having less than 3000 states. The alphabet was the english letters and the blank character. The final automaton constitutes of states that are of length 2, like 'qu' and 'xe', and on the other hand 8 and 9 symbols long states, like 'shall be' and 'there was'. This indicates that the algorithm really captures

the notion of variable context length prediction which resulted in a compact yet accurate model. Building a full Markov model in this case is impossible since it requires $|\Sigma|^L = 27^9$ states. Here we demonstrate our algorithm for cleaning corrupted text. A test text (which was taken out of the training sequence) was modified in two different ways. First by a stationary noise that altered each letter with probability **0.2**, and then the text was further modified by changing each blank to a random letter. The most probable state sequence was found via dynamic programming. The 'cleaned' observation sequence is the most probable outcome given the knowledge of the error rate. An example of such decoding for these two types of noise is shown in Fig. 4. We also applied the algorithm to intergenic

---

Original Text:

and god called the dry land earth and the gathering together of the waters called he seas and god saw that it was good and god said let the earth bring forth grass the herb yielding seed and the fruit tree yielding fruit after his kind

Noisy text (1):

and god cavsed the drxjland earth ibd shg gathervng together oj the waters cflled re seas aed god saw thctpit was good ann god said let tae earth bring forth gjasb tse hemb yielpinl peed and thesfruit tree sielxing fzuitnafter his kind

Decoded text (1):

and god caused the dry land earth and she gathering together of the waters called he sees and god saw that it was good and god said let the earth bring forth grass the memb yielding peed and the fruit tree fielding fruit after his kind

Noisy text (2):

andhgodpcilledjthesdryjlandbeasthcandmthelgatceringhlogetherjfytrezaatersoczlled xherseasaknddgodbsawwthathitqwasoqoohanwzgodcsaidhletdtheuejrthriringmforth bgrasstthexherbyieldingzseedmazdctcybfruitttreeayieldinglfruztbafherihiskind

Decoded text (2):

and god called the dry land earth and the gathering together of the altars called he seasaked god saw that it was took and god said let the earthriring forth grass the herb yielding seed and thy fruit treescielding fruit after his kind

Figure 4: Cleaning corrupted text using a Markov process with bounded memory.

---

regions of *E. coli* DNA, with $L = 20$ and $\epsilon = 0.0001$. The alphabet is: A,C,T,G. The result of the algorithm is an automaton having 80 states. The names of the states of the final automaton are depicted in Fig. 5. The performance of the model can be compared to other models, such as the HMM based model [8], by calculating the normalized log-likelihood (NLL) over unseen data. The NLL is an empirical measure of the the entropy of the source as induced by the model. The NLL of bounded memory Markov model is about the same as the one obtained by the HMM based model. Yet, the Markov model does not contain length distribution of the intergenic segments hence the overall performace of the HMM based model is slightly better. On the other hand, the HMM based model is more complicated and requires manual tuning of its architecture.

A C T G AA AC AT CA CC CT CG TA TC TT TG GA GC GT GG AAC AAT AAG
ACA ATT CAA CAC CAT CAG CCA CCT CCG CTA CTC CTT CGA CGC CGT TAT
TAG TCA TCT TTA TTG TGC GAA GAC GAT GAG GCA GTA GTC GTT GTG
GGA GGC GGT AACT CAGC CCAG CCTG CTCA TCAG TCTC TTAA TTGC
TTGG TGCC GACC GATA GAGC GGAC GGCA GGCG GGTA GGTT GGTG
CAGCC TTGCA GGCGC GGTTA

Figure 5: The states that constitute the automaton for predicting the next base of intergenic regions in *E. coli* DNA.

## 6  Conclusions and Future Research

In this paper we present a new efficient algorithm for estimating the structure and the transition probabilities of a Markov processes with bounded yet variable memory. The algorithm when applied to natural language modeling result in a compact and accurate model which captures the short term correlations. The theoretical properties of the algorithm will be described elsewhere. In fact, we can prove that a slightly different algorithm constructs a bounded memory markov process, which with arbitrary high probability, induces distributions (over $\Sigma^n$ for $n > 0$) which are very close to those induced by the 'true' Markovian source, in the sense of the KL divergence. This algorithm uses a polynomial size sample and runs in polynomial time in the relevent parameters of the problem. We are also investigating hierarchical models based on these automata which are able to capture multi-scale correlations, thus can be used to model more of the large scale structure of the natural language.

## Acknowledgment

We would like to thank Lee Giles for providing us with the software for plotting finite state machines, and Anders Krogh and David Haussler for letting us use their *E. coli* DNA data and for many helpful discussions. Y.S. would like to thank the Clore foundation for its support.

## References

[1] J.G Kemeny and J.L. Snell, *Finite Markov Chains*, Springer-Verlag 1982.

[2] Y. Freund, M. Kearns, D. Ron, R. Rubinfeld, R.E. Schapire, and L. Sellie, *Efficient Learning of Typical Finite Automata from Random Walks*, STOC-93.

[3] F. Jelinek, *Self-Organized Language Modeling for Speech Recognition*, 1985.

[4] A. Nadas, *Estimation of Probabilities in the Language Model of the IBM Speech Recognition System*, IEEE Trans. on ASSP Vol. 32 No. 4, pp. 859-861, 1984.

[5] S. Kullback, *Information Theory and Statistics*, New York: Wiley, 1959.

[6] J. Rissanen and G. G. Langdon, *Universal modeling and coding*, IEEE Trans. on Info. Theory, IT-27 (3), pp. 12-23, 1981.

[7] J. Rissanen, *Stochastic complexity and modeling*, The Ann. of Stat., 14(3), 1986.

[8] A. Krogh, S.I. Mian, and D. Haussler, *A Hidden Markov Model that finds genes in E. coli DNA*, UCSC Tech. Rep. UCSC-CRL-93-16.